# A Comparison of Dynamic Reposing and Tangent Distance for Drug Activity Prediction

**Thomas G. Dietterich**
Arris Pharmaceutical Corporation and Oregon State University
Corvallis, OR 97331-3202

**Ajay N. Jain**
Arris Pharmaceutical Corporation
385 Oyster Point Blvd., Suite 3
South San Francisco, CA 94080

**Richard H. Lathrop** and **Tomas Lozano-Perez**
Arris Pharmaceutical Corporation and MIT Artificial Intelligence Laboratory
545 Technology Square
Cambridge, MA 02139

## Abstract

In drug activity prediction (as in handwritten character recognition), the features extracted to describe a training example depend on the pose (location, orientation, etc.) of the example. In handwritten character recognition, one of the best techniques for addressing this problem is the tangent distance method of Simard, LeCun and Denker (1993). Jain, et al. (1993a; 1993b) introduce a new technique—dynamic reposing—that also addresses this problem. Dynamic reposing iteratively learns a neural network and then reposes the examples in an effort to maximize the predicted output values. New models are trained and new poses computed until models and poses converge. This paper compares dynamic reposing to the tangent distance method on the task of predicting the biological activity of musk compounds. In a 20-fold cross-validation, dynamic reposing attains 91% correct compared to 79% for the tangent distance method, 75% for a neural network with standard poses, and 75% for the nearest neighbor method.

# 1   INTRODUCTION

The task of drug activity prediction is to predict the activity of proposed drug compounds by learning from the observed activity of previously-synthesized drug compounds. Accurate drug activity prediction can save substantial time and money by focusing the efforts of chemists and biologists on the synthesis and testing of compounds whose predicted activity is high. If the requirements for highly active binding can be displayed in three dimensions, chemists can work from such displays to design new compounds having high predicted activity.

Drug molecules usually act by binding to localized sites on large receptor molecules or large enyzme molecules. One reasonable way to represent drug molecules is to capture the location of their surface in the (fixed) frame of reference of the (hypothesized) binding site. By learning constraints on the allowed location of the molecular surface (and important charged regions on the surface), a learning algorithm can form a model of the binding site that can yield accurate predictions and support drug design.

The training data for drug activity prediction consists of molecules (described by their structures, i.e., bond graphs) and measured binding activities. There are two complications that make it difficult to learn binding site models from such data.

First, the bond graph does not uniquely determine the shape of the molecule. The bond graph can be viewed as specifying a (possibly cyclic) kinematic chain which may have several internal degrees of freedom (i.e., rotatable bonds). The conformations that the graph can adopt, when it is embedded in 3-space, can be assigned energies that depend on such intramolecular interactions as the Coulomb attraction, the van der Waal's force, internal hydrogen bonds, and hydrophobic interactions. Algorithms exist for searching through the space of conformations to find local minima having low energy (these are called "conformers"). Even relatively rigid molecules may have tens or even hundreds of low energy conformers. The training data does not indicate which of these conformers is the "bioactive" one—that is, the conformer that binds to the binding site and produces the observed binding activity.

Second, even if the bioactive conformer were known, the features describing the molecular surface—because they are measured in the frame of reference of the binding site—change as the molecule rotates and translates (rigidly) in space.

Hence, if we consider feature space, each training example (bond graph) induces a family of 6-dimensional manifolds. Each manifold corresponds to one conformer as it rotates and translates (6 degrees of freedom) in space. For a classification task, a positive decision region for "active" molecules would be a region that intersects at least one manifold of each active molecule and no manifolds of any inactive molecules. Finding such a decision region is quite difficult, because the manifolds are difficult to compute.

A similar "feature manifold problem" arises in handwritten character recognition. There, the training examples are labelled handwritten digits, the features are extracted by taking a digitized gray-scale picture, and the feature values depend on the rotation, translation, and zoom of the camera with respect to the character.

We can formalize this situation as follows. Let $x_i$, $i = 1, \ldots, N$ be training examples (i.e., bond graphs or physical handwritten digits), and let $f(x_i)$ be the label associated with $x_i$ (i.e., the measured activity of the molecule or the identity of the handwritten digit). Suppose we extract $n$ real-valued features $\mathbf{V}(x_i)$ to describe object $x_i$ and then employ, for example, a multilayer sigmoid network to approximate $f(x)$ by $\hat{f}(x) = g(\mathbf{V}(x))$. This is the ordinary supervised learning task.

However, the feature manifold problem arises when the extracted features depend on the "pose" of the example. We will define the pose to be a vector $p$ of parameters that describe, for example, the rotation, translation, and conformation of a molecule or the rotation, translation, scale, and line thickness of a handwritten digit. In this case, the feature vector $\mathbf{V}(x, p)$ depends on both the example and the pose.

Within the handwritten character recognition community, several techniques have been developed for dealing with the feature manifold problem. Three existing approaches are standardized poses, the tangent-prop method, and the tangent-distance method. Jain et al. (1993a, 1993b) describe a new method—dynamic reposing—that applies supervised learning simultaneously to discover the "best" pose $p_i^*$ of each training example $x_i$ and also to learn an approximation to the unknown function $f(x)$ as $\hat{f}(x_i) = g(\mathbf{V}(x_i, p_i^*))$. In this paper, we briefly review each of these methods and then compare the performance of standardized poses, tangent distance, and dynamic reposing to the problem of predicting the activity of musk molecules.

## 2   FOUR APPROACHES TO THE FEATURE MANIFOLD PROBLEM

### 2.1   STANDARDIZED POSES

The simplest approach is to select only one of the feature vectors $\mathbf{V}(x_i, p_i)$ for each example by constructing a function, $p_i = S(x_i)$, that computes a standard pose for each object. Once $p_i$ is chosen for each example, we have the usual supervised learning task—each training example has a unique feature vector, and we can approximate $f$ by $\hat{f}(x) = g(\mathbf{V}(x, S(x)))$.

The difficulty is that $S$ can be very hard to design. In optical character recognition, $S$ typically works by computing some pose-invariant properties (e.g., principal axes of a circumscribing ellipse) of $x_i$ and then choosing $p_i$ to translate, rotate, and scale $x_i$ to give these properties standard values. Errors committed by OCR algorithms can often be traced to errors in the $S$ function, so that characters are incorrectly positioned for recognition.

In drug activity prediction, the standardizing function $S$ must guess which conformer is the bioactive conformer. This is exceedingly difficult to do without additional information (e.g., 3-D atom coordinates of the molecule bound in the binding

site as determined by x-ray crystallography). In addition, $S$ must determine the orientation of the bioactive conformers within the binding site. This is also quite difficult—the bioactive conformers must be mutually aligned so that shared potential chemical interactions (e.g., hydrogen bond donors) are superimposed.

## 2.2   TANGENT PROPAGATION

The tangent-prop approach (Simard, Victorri, LeCun, & Denker, 1992) also employs a standardizing function $S$, but it augments the learning procedure with the constraint that the output of the learned function $g(\mathbf{V}(x,p))$ should be invariant with respect to slight changes in the poses of the examples:

$$\left\| \nabla_p \, g(\mathbf{V}(x,p)) \mid_{p=S(x)} \right\| = 0,$$

where $\| \cdot \|$ indicates Euclidean norm. This constraint is incorporated by using the left-hand-side as a regularizer during backpropagation training.

Tangent-prop can be viewed as a way of focusing the learning algorithm on those input features and hidden-unit features that are invariant with respect to slight changes in pose. Without the tangent-prop constraint, the learning algorithm may identify features that "accidentally" discriminate between classes. However, tangent-prop still assumes that the standard poses are correct. This is not a safe assumption in drug activity prediction.

## 2.3   TANGENT DISTANCE

The tangent-distance approach (Simard, LeCun & Denker, 1993) is a variant of the nearest-neighbor algorithm that addresses the feature manifold problem. Ideally, the best distance metric to employ for the nearest-neighbor algorithm with feature manifolds is to compute the "manifold distance"—the point of nearest approach between two manifolds:

$$\text{manifold-dist}\,(x_1, x_2) = \min_{p_1, p_2} \, \|\mathbf{V}(x_1, p_1) - \mathbf{V}(x_2, p_2)\|.$$

This is very expensive to compute, however, because the manifolds can have highly nonlinear shapes in feature space, so the manifold distance can have many local minima.

The tangent distance is an approximation to the manifold distance. It is computed by approximating the manifold by a tangent plane in the vicinity of the standard poses. Let $\mathbf{J}_i$ be the Jacobian matrix defined by $(\mathbf{J}_i)_{jk} = \partial \mathbf{V}(x_i, p_i)_j / \partial (p_i)_k$, which gives the plane tangent to the manifold of molecule $x_i$ at pose $p_i$. The tangent distance is defined as

$$\text{tangent-dist}\,(x_1, x_2) = \min_{a,b} \|[\mathbf{V}(x_1, p_1) + \mathbf{J}_1 a] - [\mathbf{V}(x_2, p_2) + \mathbf{J}_2 b]\|,$$

where $p_1 = S(x_1)$ and $p_2 = S(x_2)$. The column vectors $a$ and $b$ give the change in the pose required to minimize the distance between the tangent planes approximating the manifolds. The values of $a$ and $b$ minimizing the right-hand side can be computed fairly quickly via gradient descent (Simard, personal communication). In practice, only poses close to $S(x_1)$ and $S(x_2)$ are considered, but this provides

more opportunity for objects belonging to the same class to adopt poses that make them more similar to each other.

In experiments with handwritten digits, Simard, LeCun, and Denker (1993) found that tangent distance gave the best performance of these three methods.

## 2.4  DYNAMIC REPOSING

All of the preceding methods can be viewed as attempts to make the final predicted output $\hat{f}(x)$ invariant with respect to changes in pose. Standard poses do this by not permitting poses to change. Tangent-prop adds a local invariance constraint. Tangent distance enforces a somewhat less local invariance constraint.

In dynamic reposing, we make $\hat{f}$ invariant by defining it to be the maximum value (taken over all poses $p$) of an auxiliary function $g$:

$$\hat{f}(x) = \max_p \ g(\mathbf{V}(x, p)).$$

The function $g$ will be the function learned by the neural network.

Before we consider how $g$ is learned, let us first consider how it can be used to predict the activity of a new molecule $x'$. To compute $\hat{f}(x')$, we must find the pose $p'^*$ that maximizes $g(\mathbf{V}(x', p'^*))$. We can do this by performing a gradient ascent starting from the standard pose $S(x)$ and moving in the direction of the gradient of $g$ with respect to the pose: $\nabla_{p'} g(\mathbf{V}(x', p'))$.

This process has an important physical analog in drug activity prediction. If $x'$ is a new molecule and $g$ is a learned model of the binding site, then by varying the pose $p'$ we are imitating the process by which the molecule chooses a low-energy conformation and rotates and translates to "dock" with the binding site.

In handwritten character recognition, this would be the dual of a deformable template model: the template ($g$) is held fixed, while the example is deformed (by rotation, translation, and scaling) to find the best fit to the template.

The function $g$ is learned iteratively from a growing pool of feature vectors. Initially, the pool contains only the feature vectors for the standard poses of the training examples (actually, we start with one standard pose of each low energy conformation of each training example). In iteration $j$, we apply backpropagation to learn hypothesis $g_j$ from selected feature vectors drawn from the pool. For each molecule, one feature vector is selected by performing a forward propagation (i.e., computing $g(\mathbf{V}(x_i, p_i))$) of all feature vectors of that molecule and selecting the one giving the highest predicted activity for that molecule.

After learning $g_j$, we then compute for each conformer the pose $p_i^{j+1}$ that maximizes $g_j(\mathbf{V}(x_i, p))$:

$$p_i^{j+1} = \underset{p}{\mathrm{argmax}} \ g_j(\mathbf{V}(x_i, p)).$$

From the chemical perspective, we permit each of the molecules to "dock" to the current model $g_j$ of the binding site.

The feature vectors $\mathbf{V}(x_i, p_i^{j+1})$ corresponding to these poses are added to the pool of poses, and a new hypothesis $g_{j+1}$ is learned. This process iterates until the poses

cease to change. Note that this algorithm is analogous to the EM procedure (Redner & Walker, 1984) in that we accomplish the simultaneous optimization of $g$ and the poses $\{p_i\}$ by conducting a series of separate optimizations of $g$ (holding $\{p_i\}$ fixed) and $\{p_i\}$ (holding $g$ fixed).

We believe the power of dynamic reposing results from its ability to identify the features that are critical for discriminating active from inactive molecules. In the initial, standard poses, a learning algorithm is likely to find features that "accidentally" discriminate actives from inactives. However, during the reposing process, inactive molecules will be able to reorient themselves to resemble active molecules with respect to these features. In the next iteration, the learning algorithm is therefore forced to choose better features for discrimination.

Moreover, during reposing, the active molecules are able to reorient themselves so that they become *more* similar to each other *with respect to the features judged to be important* in the previous iteration. In subsequent iterations, the learning algorithm can "tighten" its criteria for recognizing active molecules.

In the initial, standard poses, the molecules are posed so that they resemble each other along all features more-or-less equally. At convergence, the active molecules have changed pose so that they only resemble each other along the features important for discrimination.

# 3   AN EXPERIMENTAL COMPARISON

## 3.1   MUSK ACTIVITY PREDICTION

We compared dynamic reposing with the tangent distance and standard pose methods on the task of musk odor prediction. The problem of musk odor prediction has been the focus of many modeling efforts (e.g., Bersuker, et al., 1991; Fehr, et al., 1989; Narvaez, Lavine & Jurs, 1986). Musk odor is a specific and clearly identifiable sensation, although the mechanisms underlying it are poorly understood. Musk odor is determined almost entirely by steric (i.e., "molecular shape") effects (Ohloff, 1986). The addition or deletion of a single methyl group can convert an odorless compound into a strong musk. Musk molecules are similar in size and composition to many kinds of drug molecules.

We studied a set of 102 diverse structures that were collected from published studies (Narvaez, Lavine & Jurs, 1986; Bersuker, et al., 1991; Ohloff, 1986; Fehr, et al., 1989). The data set contained 39 aromatic, oxygen-containing molecules with musk odor and 63 homologs that lacked musk odor. Each molecule was conformationally searched to identify low energy conformations. The final data set contained 6,953 conformations of the 102 molecules (for full details of this data set, see Jain, et al., 1993a). Each of these conformations was placed into a starting pose via a hand-written $S$ function. We then applied nearest neighbor with Euclidean distance, nearest neighbor with the tangent distance, a feed-forward network without reposing, and a feed-forward network with the dynamic reposing method. For dynamic reposing, five iterations of reposing were sufficient for convergence. The time required to compute the tangent distances far exceeds the computation times of the other algorithms. To make the tangent distance computations feasible, we only

Table 1: Results of 20-fold cross-validation on 102 musk molecules.

| Method | Percent Correct |
|---|---|
| Nearest neighbor (Euclidean distance) | 75 |
| Neural network (standard poses) | 75 |
| Nearest neighbor (Tangent distance) | 79 |
| Neural network (dynamic reposing) | 91 |

Table 2: Neural network cross-class predictions (percent correct)

| N | 13 | 21 | 27 | 14 |
|---|---|---|---|---|
| Molecular class: | 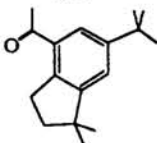 | 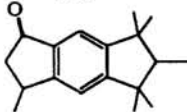 | 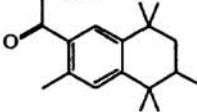 | 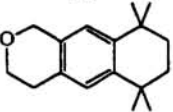 |
| Standard poses | 85 | 76 | 74 | 57 |
| Dynamic reposing | 100 | 90 | 85 | 71 |

computed the tangent distance for the 200 neighbors that were nearest in Euclidean distance. Experiments with a subset of the molecules showed that this heuristic introduced no error on that subset.

Table 1 shows the results of a 20-fold cross-validation of all four methods. The tangent distance method does show improvement with respect to a standard neural network approach (and with respect to the standard nearest neighbor method). However, the dynamic reposing method outperforms the other two methods substantially.

An important test for drug activity prediction methods is to predict the activity of molecules whose molecular structure (i.e., bond graph) is substantially different from the molecules in the training set. A weakness of many existing methods for drug activity prediction (Hansch & Fujita, 1964; Hansch, 1973) is that they rely on the assumption that all molecules in the training and test data sets share a common structural skeleton. Because our representation for molecules concerns itself only with the surface of the molecule, we should not suffer from this problem. Table 2 shows four structural classes of molecules and the results of "class holdout" experiments in which all molecules of a given class were excluded from the training set and then predicted. Cross-class predictions from standard poses are not particularly good. However, with dynamic reposing, we obtain excellent cross-class predictions. This demonstrates the ability of dynamic reposing to identify the critical discriminating features. Note that the accuracy of the predictions generally is determined by the size of the training set (i.e., as more molecules are withheld, performance drops). The exception to this is the right-most class, where the local geometry of the oxygen atom is substantially different from the other three classes.

# 4   CONCLUDING REMARKS

The "feature manifold problem" arises in many application tasks, including drug activity prediction and handwritten character recognition. A new method, dynamic reposing, exhibits performance superior to the best existing method, tangent distance, and to other standard methods on the problem of musk activity prediction. In addition to producing more accurate predictions, dynamic reposing results in a learned binding site model that can guide the design of new drug molecules. Jain, et al., (1993a) shows a method for visualizing the learned model in the context of a given molecule and demonstrates how the model can be applied to guide drug design. Jain, et al., (1993b) compares the method to other state-of-the-art methods for drug activity prediction and shows that feed-forward networks with dynamic reposing are substantially superior on two steroid binding tasks. The method is currently being applied at Arris Pharmaceutical Corporation to aid the development of new pharmaceutical compounds.

**Acknowledgements**

Many people made contributions to this project. The authors thank Barr Bauer, John Burns, David Chapman, Roger Critchlow, Brad Katz, Kimberle Koile, John Park, Mike Ross, Teresa Webster, and George Whitesides for their efforts.

**References**

Bersuker, I. B., Dimoglo, A. S., Yu. Gorbachov, M., Vlad, P. F., Pesaro, M. (1991). *New Journal of Chemistry, 15*, 307.

Fehr, C., Galindo, J., Haubrichs, R., Perret, R. (1989). *Helv. Chim. Acta, 72*, 1537.

Hansch, C. (1973). In C. J. Cavallito (Ed.), *Structure-Activity Relationships*. Oxford: Pergamon.

Hansch, C., Fujita, T. (1964). *J. Am. Chem. Soc., 86*, 1616.

Jain, A. N., Dietterich, T. G., Lathrop, R. H., Chapman, D., Critchlow, R. E., Bauer, B. E., Webster, T. A., Lozano-Perez, T. (1993a). A shape-based method for molecular design with adaptive alignment and conformational selection. Submitted.

Jain, A., Koile, K., Bauer, B., Chapman, D. (1993b). Compass: A 3D QSAR method. Performance comparisons on a steroid benchmark. Submitted.

Narvaez, J. N., Lavine, B. K., Jurs, P. C. (1986). *Chemical Senses, 11*, 145–156.

Ohloff, G. (1986). Chemistry of odor stimuli. *Experientia, 42*, 271.

Redner, R. A., Walker, H. F. (1984). Mixture densities, maximum likelihood, and the EM algorithm. *SIAM Review, 26* (2) 195–239.

Simard, P. Victorri, B., Le Cun, Y. Denker, J. (1992). Tangent Prop—A formalism for specifying selected invariances in an adaptive network. In Moody, J. E., Hanson, S. J., Lippmann, R. P. (Eds.) *Advances in Neural Information Processing Systems 4*. San Mateo, CA: Morgan Kaufmann. 895–903.

Simard, P. Le Cun, Y., Denker, J. (1993). Efficient pattern recognition using a new transformation distance. In Hanson, S. J., Cowan, J. D., Giles, C. L. (Eds.) *Advances in Neural Information Processing Systems 5*, San Mateo, CA: Morgan Kaufmann. 50–58.